# Joint Cascade Optimization Using a Product of Boosted Classifiers

**Leonidas Lefakis**

Idiap Research Institute

Martigny, Switzerland

`leonidas.lefakis@idiap.ch`

**François Fleuret**

Idiap Research Institute

Martigny, Switzerland

`francois.fleuret@idiap.ch`

## Abstract

The standard strategy for efficient object detection consists of building a cascade composed of several binary classifiers. The detection process takes the form of a lazy evaluation of the conjunction of the responses of these classifiers, and concentrates the computation on difficult parts of the image which cannot be trivially rejected.

We introduce a novel algorithm to construct jointly the classifiers of such a cascade, which interprets the response of a classifier as the probability of a positive prediction, and the overall response of the cascade as the probability that all the predictions are positive. From this noisy-AND model, we derive a consistent loss and a Boosting procedure to optimize that global probability on the training set.

Such a joint learning allows the individual predictors to focus on a more restricted modeling problem, and improves the performance compared to a standard cascade. We demonstrate the efficiency of this approach on face and pedestrian detection with standard data-sets and comparisons with reference baselines.

## 1 Introduction

Object detection remains one of the core objectives of computer vision, either as an objective *per se*, for instance for automatic focusing on faces in digital cameras, or as means to get high-level understanding of natural scenes for robotics and image retrieval.

The standard strategy which has emerged for detecting objects of reasonable complexity such as faces is the so-called "sliding-window" approach. It consists of visiting all locations and scales in the scene to be parsed, and for any such pose, evaluating a two-class predictor which computes if the object of interest is visible there.

The computational cost of such approaches is controlled traditionally with a cascade, that is a succession of classifiers, each one being evaluated only if the previous ones in the sequence have not already rejected the candidate location. Such an architecture concentrates the computation on difficult parts of the global image to be processed, and reduces tremendously the overall computational effort.

In its original form, this approach constructs classifiers one after another during training, each one from examples which have not been rejected by the previous ones. While very successful, this technique suffers from three main practical drawbacks. The first one is the need for a very large number of negative samples, so that enough samples are available to train any one of the classifiers. The second drawback is the necessity to define as many thresholds as there are levels in the cascade. This second step may seem innocuous, but in practice is a serious difficulty, requiring additional validation data. Finally the third drawback is the inability of a standard cascade to properly exploit

the trade-off between the different levels. A response marginally below threshold at a certain level is enough to reject a sample, even if classifiers at other levels have strong responses.

At a more conceptual level, standard training for cascades does not allow the classifiers to exploit their joint modeling: Each classifier is trained as if it has to do the job alone, without having the opportunity to properly balance its own modeling effort and that of the other classifiers.

The novel approach we propose here is a *joint learning* of the classifiers constituting a cascade. We interpret the individual responses of the classifiers as probabilities of responding positively, and define the overall response of the cascade as the probability of all the classifiers responding positively under an assumption of independence. Instead of training classifiers successively, we directly minimize a loss taking into account this global response. This noisy-AND model leads to a very simple criterion for a new Boosting procedure, which improves all the classifiers symmetrically on the positive samples, and focuses on improving the classifier with the best response on every negative sample.

We demonstrate the efficiency of this technique for face and pedestrian detection. Experiments show that this joint cascade learning requires far less negative training examples, and achieves performance better than standard cascades without the need for intensive bootstrapping. At the computational level, we propose to optimally permute the order of the classifiers during the evaluation to reduce the overall number of evaluated classifiers, and show that such optimization allows for better error rates at similar computational costs.

## 2 Related works

A number of methods have been proposed over the years to control the computational cost of machine-learning based object detection. The idea common to these approaches is to rely on a form of adaptive testing : only candidates which cannot be trivially rejected as not being the object of interest will require heavy computation. In practice the majority of the candidates will be rejected with a very coarse criterion, hence requiring very low computation.

### 2.1 Reducing object detection computational cost

Heisele et al. [1] propose a hierarchy of linear Support Vector Machines, each trained on images of increasing resolution, to weed out background patches, followed by a final computationally intensive polynomial SVM. In [2] and [3], the authors use an hierarchy of respectively two and three Support Vector Machines of increasing complexity. Graf et al. [4] introduced the parallel support vector machine which creates a filtering process by combining layers of parallel SVMs, each trained using the support vectors of classifiers in the previous layer.

Fleuret and Geman [5] introduce a hierarchy of classifiers dedicated to positive populations with geometrical poses of decreasing randomness. This approach generalizes the cascade to more complex pose spaces, but as for cascades, trains the classifiers separately.

Recently, a number of scanning alternatives to sliding window have also been introduced. In [6] a branch and bound approach is utilized during scanning, while in [7] a divide and conquer approach is proposed, wherein regions in the image are either accepted or rejected as a whole or split and further processed. Feature-centric approaches is proposed by the authors in [8] and [9].

The most popular approach however, for both its conceptual simplicity and practical efficiency, is the attentional cascade proposed by Viola and Jones [10]. Following this seminal paper, cascades have been used in a variety of problems [11, 12, 13].

### 2.2 Improving attentional cascades

In recent years approaches have been proposed that address some of the issues we list in the introduction. In [14] the authors train a cascade with a global performance criteria and a single set of parameters common to all stages. In [15] the authors address the asymmetric nature of the stage goals via a biased minimax probability machine, while in [16] the authors formulate the stage goals as a constrained optimization problem. In [17] a alternate boosting method dubbed FloatBoost is proposed. It allows for backtracking and removing weak classifiers which no longer contribute.

Table 1: Notation

$(x_n, y_n), \ n = 1, \ldots, N$, training examples.

$\quad K$ number of levels in the cascade.

$\quad f_k(x)$ non-thresholded response of classifier $k$. During training, $f_k^t(x)$ stands for that response after $t$ steps of Boosting.

$\quad p_k(x) = \frac{1}{1+\exp(-f_k(x))}$ probability of classifier $k$ to response positively on $x$. During training, $p_k^t(x)$ stands for the same value after $t$ steps of Boosting, computed from $f_k^t(x)$.

$\quad p(x) = \prod_k p_k(x)$ posterior probability of sample $x$ to be positive, as estimated jointly by all the classifiers of the cascade. During training, $p^t(x)$ is that value after only $t$ steps of Boosting, computed from the $p_k^t(x)$.

Sochman and Matas [18] presented a Boosting algorithm based on sequential probability ratio tests, minimizing the average evaluation time subject to upper bounds on the false negative and false positive rates. A general framework for probabilistic boosting trees (of which cascades are a degenerated case) was proposed in [19]. In all these methods however, a set of free parameters concerning detection and false alarm performances must be set during training. As will be seen, our method is capable of postponing any decisions concerning performance goals until after training.

The authors in [20] use the output of each stage as an initial weak classifier of the boosting classifier in the next stage. This allows the cascade to retain information between stages. However this approach only constitutes a backward view of the cascade. No information concerning the future performance of the cascade is available to each stage. In [21] sample traces are utilized to keep track of the performance of the cascade on the training data, and thresholds are picked after the cascade training is finished. This allows for reordering of cascade stages. However besides a validation set, a large number of negative examples must also be bootstrapped not only during the training phase, but also during the post-processing step of threshold and order calibration. Furthermore, different learning targets are used in the learning and calibration phases.

To our knowledge, very little work has been done on the joint optimization of the cascaded stages. In [22] the authors attempt to jointly optimize a cascade of SVMs. As can be seen, a cascade effectively performs an AND operation over the data, enforcing that a positive example passes all stages; and that a negative example be rejected by at least one stage. In order to simulate this behavior, the authors attempt to minimize the maximum hinge loss over the SVMs for the positive examples, and to minimize the product of the hinge losses for the negative examples. An approximate solution to this formulation is found via cyclic optimization. In [23] the authors present a method similar to ours, jointly optimizing a cascade using the product of the output of individual logistic regression base classifiers. Their method attempts to find the MAP-estimate of the optimal classifier weights using cyclic coordinate descent. As is the case with the work in [22], the authors consider the ordering of the stages a priori fixed.

## 3  Method

Our approach can be interpreted as a noisy-AND: The classifiers in the cascade produce stochastic Boolean predictions, conditionally independent given the signal to classify. We define the global response of the cascade as the probability that all these predictions are positive.

This can be interpreted as if we were first computing from the signal $x$, for each classifier in the cascade, a probability $p_k(x)$, and defining the response of the cascade as the probability that $K$ independent Bernoulli variables of parameters $p_1(x), \ldots, p_K(x)$ would all be equal to 1. Such a criterion takes naturally into account the confidence of individual classifiers in the final response, and introduces an additional non-linearity in the decision function.

This approach is related to the noisy-OR proposed in [24] for multi-view object detection. However, their approach aims at decomposing a complex population into a collection of homogeneous populations, while our objective is to speed up the computation for the detection of a homogeneous

population. In some sense the noisy-OR they propose and the noisy-AND we use for training are addressing dual objectives.

## 3.1 Formalization

Let $f_k(x)$ stand for the non-thresholded response of the classifier at level $k$ of the cascade. We define

$$p_k(x) = \frac{1}{1 + \exp(-f_k(x))} \tag{1}$$

as the probabilistic interpretation of the deterministic output of classifier $k$.

From that, we define the final output of the cascade as the probability that all classifiers make positive predictions, under the assumption that they are conditionally independent, given $x$

$$p(x) = \prod_{k=1}^{K} p_k(x). \tag{2}$$

In the ideal Boolean case, an example $x$ will be classified as positive if and only if all classifiers classify it as such. Conversely the example will be classified as negative if $p_k(x) = 0$ for at least one $k$. This is consistent with the AND nature of the cascade. Of course due to the product, the final classifier is able to make probabilistic predictions rather than solely hard ones as in [22].

## 3.2 Joint Boosting

Let

$$(x_n, y_n) \in \mathbb{R}^d \times \{0, 1\}, \quad n = 1, \dots, N \tag{3}$$

denote a training set. In order to train our cascade we consider the maximization of the joint maximum log likelihood of the data:

$$J = \log \prod_n p(x_n)^{y_n} (1 - p(x_n))^{1-y_n}. \tag{4}$$

At each round $t$ we sequentially visit each classifier and add a weak learner which locally minimizes $J$ the most. If $p^t(x)$ denotes the overall response of the cascade after having added $t$ weak learners in each classifier, and $p_k^t(x)$ denotes the response of classifier $k$ at that point – hence a function the response of classifier $k$ at step $t$, $f_k^t(x)$ – the score to maximize to select a weak learner $h_t^k(x_n)$ is:

$$\sum_n w_n^{k,t} h_t^k(x_n) \tag{5}$$

with

$$w_n^{k,t} = \frac{\partial J}{\partial f_k(x_n)} = \frac{y_n - p^t(x_n)}{1 - p^t(x_n)} (1 - p_k^t(x_n)). \tag{6}$$

It should be noted that in this formulation, the weight $w_n^{k,t}$ are signed, and these assigned to negative examples are negative.

In the case of a positive example $x_n$ this simplifies to $w_n^{k,t} = 1 - p_k^t(x_n)$ and thus this criterion pushes every classifier in the cascade to maximize the response on positive samples, irrespective of the performance of the overall cascade.

In the case of a negative example however, the weight update rule becomes $w_n^{k,t} = \frac{-p^t(x_n)}{1-p^t(x_n)}(1 - p_k^t(x_n))$, each classifier in the cascade is then passed information regarding the overall performance via the term $\frac{-p^t(x_n)}{1-p^t(x_n)}$. If the cascade is already rejecting the negative example, then this term becomes 0 and the classifier ignores its performance on the specific example. On the other hand, if the cascade is performing poorly, then the term becomes increasingly large and the classifiers put large weights on that example.

Furthermore, due to the term $1 - p_k^t(x_n)$, each classifier puts larger weight on negative examples that it is already performing well on, effectively partitioning the space of negative examples.

The weights of the weak-learners can not be computed in a close formed as for AdaBoost and are estimated through a numerical line-search.

### 3.3 Exponential variant

To assess if the asymptotic behavior of the loss – which is similar in spirit to the logistic one – is critical or not in the performance, we also experimented the minimization of the exponential error of the output.

This translates to the minimization of the cost function :

$$J^{exp} = \sum_n \left( \frac{1 - p(x_n)}{p(x_n)} \right)^{2y_n - 1} \tag{7}$$

and leads to the following expression for the sample weights during Boosting:

$$w_n^{k,t} = \frac{p_k^t(x_n) - 1}{p^t(x_n)} \tag{8}$$

for the positive samples and

$$w_n^{k,t} = \frac{\left(1 - p_k^t(x_n)\right) p^t(x_n)}{\left(1 - p^t(x_n)\right)^2} \tag{9}$$

for the negative ones.

Such a weighting strongly penalizes outliers in the training set, in a manner similar to Adaboost's exponential loss.

## 4 Experiments

### 4.1 Implementation Details

We comparatively evaluate the proposed cascade framework on two data-sets. In [10] the authors present an initial comparison between their cascade framework and an AdaBoost classifier on the CMU-MIT data-set. They train the monolithic classifier for 200 rounds and compare it against a simple cascade containing ten stages, each with 20 weak learners. As cascade architecture plays an important role in the final performance of the cascade, and in order to avoid any issues in the comparison pertaining to architectural designs, we keep this structure and evaluate both the proposed cascade and the Viola and Jones cascade, using this architecture. The monolithic classifier is similarly trained for 200 rounds. During the training, the thresholds for each stage in the Viola and Jones cascade are set to achieve a 99.5% detection rate.

As pointed out, our approach does not make use of a validation set, nor uses bootstrapping during training. We experimented with bootstrapping a fixed number $M$ of negative examples at fixed intervals, similar to [21] and attained higher performance than the one presented here. However it was found that training, was highly sensitive to the choice of $M$ and that furthermore this choice of $M$ was application specific.

We tested three versions of our JointCascade approach: **JointCascade** is the algorithm described in § 3.2, **JointCascade Augmented** is the same, but is trained with as many negative examples as the total number used by the Viola and Jones cascade, and **JointCascade Exponential** uses the same number of negative samples as the basic setting, but uses the exponential version of the loss described in § 3.3.

### 4.2 Data-Sets

#### 4.2.1 Pedestrians

For pedestrian detection we use the INRIA pedestrian data-set [25], which contains pedestrian images of various poses with high variance concerning background and lighting. The training set consists of 1239 images of pedestrians as positive examples, and 12180 negative examples, mined from 1218 pedestrian-free images. Of these we keep 900 images for training (together with their mirror images, for a total of 1800) and 9000 negative examples. The remaining images in the original training set are put aside to be used as a validation set by the Viola and Jones cascade.

As in [25] we utilize a histogram of oriented gradient to describe each image. The reader is referred to this article for implementation details of the descriptor.

The trained classifiers are then tested on a test set composed of 1126 images of pedestrians and 18120 non-pedestrian images.

### 4.2.2 Faces

For faces, we evaluate against the CMU+MIT data-set of frontal faces. We utilize the Haar-like wavelet features introduced in [10], however, for performance reasons, we sub-sample 2000 of these features at each round to be used for training.

For training we use the same data-set as that used by Viola and Jones consisting of 4916 images of faces. Of these we use 4000 (plus their mirror images) for training and set apart a further 916 (plus mirror images) for use as the validation set needed by the classical cascade approach. The negative portion of the training set is comprised of 10000 non-face images, mined randomly from non-face containing images.

In order to test the trained classifiers, we extract the 507 faces in the data-set and scale-normalize to 24x24 images, a further 12700 non-face image patches are extracted from the background of the images in the data-set. We do not perform scale search, nor do we use any form of post-processing.

### 4.2.3 Bootstrap Images

As, during training, the Viola and Jones cascade needs to bootstrap false positive examples after each stage, we randomly mine a data-set of approximately 7000 images from the web. These images have been manually inspected to ensure that they do not contain either faces or pedestrians. These images are used for bootstrapping in both sets of experiments.

### 4.3 Error rate

The evaluation on the face data-set can be seen in Figure 1. The plotted lines represent the ROC curves for the evaluated methods. The proposed methods are able to reach a level of performance on par with the Viola and Jones cascade, without the need for a validation set or bootstrapping. The log-likelihood version of our method, performs slightly better than the exponential error version.

The ROC curves for the pedestrian detection task can be seen in Figure 2. The log-likelihood version of our method significantly outperforms the Viola and Jones Cascade. The exponential error version is again slightly worse than the log-likelihood version, however this too outperforms the classical approach. Finally, as can be seen, augmenting the training data for the proposed method, leads to further improvement.

The results on the two data-sets show that the proposed methods are capable of performing on par or better than the Viola and Jones cascade, while avoiding the need for a validation set or for bootstrapping. This lack of a need for bootstrapping, further means that the training time needed is considerably smaller than in the case of the classical cascade.

### 4.4 Optimization of the evaluation order

As stated, one of the main motivations for using cascades is speed. We compare the average number of stages visited per negative example for the various methods presented.

Typically in cascade training, the thresholds and orders of the various stages must be determined during training, either by setting them in an *ad hoc* manner or by using one of the optimization schemes of the many proposed. In our case however, any decision concerning the thresholds as well as the ordering of the stages can be postponed till after training. It is easy to derive for any given detection goal, a relevant threshold $\theta$ on the overall cascade responce. Thus we ask that $p(x_n) > \theta$, for an image patch to be accepted as positive. Subsequently the image patch will be rejected if the product of any subset of strong classifiers has a value smaller than $\theta$.

Based on this we use a greedy method to evaluate, using the original training set, the optimal order of classifiers as follows : Originally we chose as the first stage in our cascade, the classifier whose

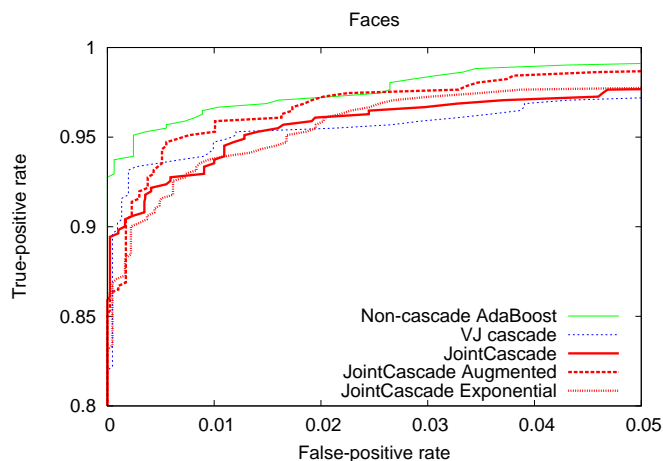

Figure 1: True-positive rate vs. false-positive rate on the face data-set for the methods proposed, AdaBoost and the Viola and Jones type cascade. The JointCascade variants are described in § 4.1. At any true-positive rate above 95%, all three methods perform better than the standard cascade. This is a particularly good result for the basic JointCascade which does not use bootstrapping during training, which would seem to be critical for such conservative regimes.

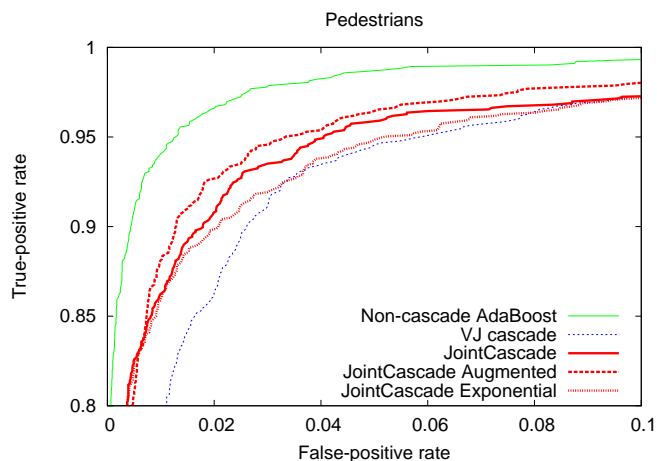

Figure 2: True-positive rate vs. false-positive rate on the pedestrian data-set for the methods proposed, AdaBoost and the Viola and Jones type cascade. All three JointCascade methods outperform the standard cascade, for regions of the false positive rate which are of practical use.

Table 2: Average number of classifiers evaluated on a sample, for each method and different true-positive rates, on the two data-sets. As expected, the computational load increases with the accuracy. The JointCascade variants require marginally more operations at a fixed rate on the pedestrian population, and marginally less on the faces except at very conservative rates. This is an especially good result, given their lower false-positive rates, which should induce more computation on average.

| TP | Computational cost (faces) | | | | Computational cost (pedestrians) | | | |
|---|---|---|---|---|---|---|---|---|
| | VJ | JointCascade | JointCascade Augmented | JointCascade Exponential | VJ | JointCascade | JointCascade Augmented | JointCascade Exponential |
| 95% | 1.35 | 1.49 | 1.62 | 1.69 | 2.27 | 2.58 | 2.66 | 2.93 |
| 90% | 1.21 | 1.18 | 1.31 | 1.25 | 1.93 | 2.04 | 1.94 | 2.21 |
| 86% | 1.13 | 1.09 | 1.18 | 1.11 | 1.56 | 1.79 | 1.71 | 1.81 |
| 82% | 1.10 | 1.04 | 1.12 | 1.07 | 1.38 | 1.49 | 1.59 | 1.52 |
| 78% | 1.07 | 1.03 | 1.09 | 1.04 | 1.30 | 1.37 | 1.48 | 1.39 |

reponse is smaller than $\theta$ for the largest number of negative examples. We then iteratively add to the order of the cascade, that classifier which leads to a response smaller than $\theta$ for the most negative examples, when multiplied with the aggregated response of the stages already ordered in the cascade.

As stated this ordering of the cascade stages is computed using the training set. We then measure the speed of our ordered cascade on the same test sets as above, as shown on Table 2. As can be seen, in the case of the face dataset, in almost all cases our approach is actually faster during scanning than the classical Viola and Jones approach. When the augmented dataset is used however this speed advantage is lost, there is a thus a trade-off between performance and speed, as is to be expected. The speed of our JointCascade approach on the pedestrian data-set is marginally worst than that of Viola and Jones, which is due to the lower false-positive rates.

## 5 Conclusion

We have presented a new criterion to train a cascade of classifiers in a joint manner. This approach has a clear probabilistic interpretation as a noisy-AND, and leads to a global decision criterion which avoids thresholding classifiers individually, and can exploit independence in the classifier response amplitudes.

This method avoids the need for picking multiple thresholds and the requirement for additional validation data. It allows to easily fix the final performance without the need for re-training. Finally, we have demonstrated that it reaches state-of-the-art performance on standard data sets, without the need for bootstrapping.

This approach is very promising as a general framework to build adaptive detection techniques. It could easily be extended to hierarchical approaches instead of simple cascade, hence could be used for latent poses richer than location and scale.

Finally, the reduction of the computational cost itself could be addressed in a more explicit manner than the optimization of the order presented in § 4.4. We are investigating a dynamic approach where the same criterion is used to allocate weak learners adaptively among the classifiers. This could be combined with a loss function explicitly estimating the expected computation cost of detection, hence providing an incentive for early rejection of more samples in the cascade.

**Acknowledgments**

We thank the anonymous reviewers for their helpful comments. This work was supported by the European Community's Seventh Framework Programme FP7 - Challenge 2 - Cognitive Systems, Interaction, Robotics - under grant agreement No 247022 - MASH.

# References

[1] B. Heisele, T. Serre, S. Prentice, and T. Poggio. Hierarchical classification and feature reduction for fast face detection with support vector machines. *Pattern Recognition Letters*, 36(9):2007–2017, 2003.

[2] Hedi Harzallah, Frédéric Jurie, and Cordelia Schmid. Combining efficient object localization and image classification. In *International Conference on Computer Vision*, pages 237–244, 2009.

[3] A. Vedaldi, V. Gulshan, M. Varma, and A. Zisserman. Multiple kernels for object detection. In *International Conference on Computer Vision*, pages 606–613, 2009.

[4] Hans Peter Graf, Eric Cosatto, Léon Bottou, Igor Dourdanovic, and Vladimir Vapnik. Parallel support vector machines: The cascade svm. In *Neural Information Processing Systems*, pages 521–528, 2005.

[5] F. Fleuret and D. Geman. Coarse-to-fine face detection. *International Journal of Computer Vision*, 41(1/2):85–107, 2001.

[6] Christopher H. Lampert, M. B. Blaschko, and Thomas Hofmann. Beyond sliding windows: Object localization by efficient subwindow search. In *Conference on Computer Vision and Pattern Recognition*, pages 1–8, 2008.

[7] Christoph H. Lampert. An efficient divide-and-conquer cascade for nonlinear object detection. In *Conference on Computer Vision and Pattern Recognition*, pages 1022–1029, 2010.

[8] Henry Schneiderman. Feature-centric evaluation for efficient cascaded object detection. In *Conference on Computer Vision and Pattern Recognition*, pages 29–36, 2004.

[9] A. Lehmann, B. Leibe, and L. Van Gool. Feature-centric efficient subwindow search. In *International Conference on Computer Vision*, pages 940–947, 2009.

[10] Paul Viola and Michael Jones. Rapid object detection using a boosted cascade of simple features. In *Conference on Computer Vision and Pattern Recognition*, pages 511–518, 2001.

[11] Owen T. Carmichael and Martial Hebert. Shape-based recognition of wiry objects. In *Conference on Computer Vision and Pattern Recognition*, pages 401–408, 2003.

[12] Qiang Zhu, Shai Avidan, Mei chen Yeh, and Kwang ting Cheng. Fast human detection using a cascade of histograms of oriented gradients. In *Conference on Computer Vision and Pattern Recognition*, pages 1491–1498, 2006.

[13] Geremy Heitz, Stephen Gould, Ashutosh Saxena, and Daphne Koller. Cascaded classification models: Combining models for holistic scene understanding. In *Neural Information Processing Systems*, pages 641–648, 2009.

[14] S. Charles Brubaker, Jianxin Wu, Jie Sun, Matthew D. Mullin, and James M. Rehg. On the design of cascades of boosted ensembles for face detection. *International Journal of Computer Vision*, 77(1-3):65–86, 2008.

[15] Kaizhu Huang, Haiqin Yang, Irwin King, and Michael R. Lyu. Learning classifiers from imbalanced data based on biased minimax probability machine. In *Conference on Computer Vision and Pattern Recognition*, pages 558–563, 2004.

[16] J. Wu, S. C. Brubaker, M. D. Mullin, and J. M. Rehg. Fast asymmetric learning for cascade face detection. *IEEE Transactions on Pattern Analysis and Machine Intelligence*, 30:369–382, 2008.

[17] Stan Z. Li and ZhenQiu Zhang. FloatBoost learning and statistical face detection. *IEEE Transactions on Pattern Analysis and Machine Intelligence*, 26(9), 2004.

[18] Jan Sochman and Jiri Matas. Waldboost " learning for time constrained sequential detection. In *Conference on Computer Vision and Pattern Recognition*, pages 150–156, 2005.

[19] Zhuowen Tu. Probabilistic boosting-tree: Learning discriminative models for classification, recognition, and clustering. In *International Conference on Computer Vision*, pages 1589–1596, 2005.

[20] Rong Xiao, Long Zhu, and HongJiang Zhang. Boosting chain learning for object detection. In *International Conference on Computer Vision*, pages 709–715, 2003.

[21] Lubomir Bourdev and Jonathan Brandt. Robust object detection via soft cascade. In *Conference on Computer Vision and Pattern Recognition*, pages 236–243, 2005.

[22] M. Murat Dundar and Jinbo Bi. Joint optimization of cascaded classifiers for computer aided detection. In *Conference on Computer Vision and Pattern Recognition*, pages 1–8, 2007.

[23] V. C. Raykar, B. Krishnapuram, and S. Yu. Designing efficient cascaded classifiers: Tradeoff between accuracy and cost. In *Conference on Knowledge Discovery and Data Mining*, 2010.

[24] Tae-Kyun Kim and Roberto Cipolla. MCBoost: Multiple classifier boosting for perceptual co-clustering of images and visual features. In *Neural Information Processing Systems*, pages 841–856, 2008.

[25] N. Dalal and B. Triggs. Histograms of oriented gradients for human detection. In *Conference on Computer Vision and Pattern Recognition*, pages 886–893, 2005.

